# A Scalable Machine Learning Approach to Go

**Lin Wu and Pierre Baldi**
School of Information and Computer Sciences
University of California, Irvine
Irvine, CA 92697-3435
lwu,pfbaldi@ics.uci.edu

## Abstract

Go is an ancient board game that poses unique opportunities and challenges for AI and machine learning. Here we develop a machine learning approach to Go, and related board games, focusing primarily on the problem of learning a good evaluation function in a scalable way. Scalability is essential at multiple levels, from the library of local tactical patterns, to the integration of patterns across the board, to the size of the board itself. The system we propose is capable of automatically learning the propensity of local patterns from a library of games. Propensity and other local tactical information are fed into a recursive neural network, derived from a Bayesian network architecture. The network integrates local information across the board and produces local outputs that represent local territory ownership probabilities. The aggregation of these probabilities provides an effective strategic evaluation function that is an estimate of the expected area at the end (or at other stages) of the game. Local area targets for training can be derived from datasets of human games. A system trained using only $9 \times 9$ amateur game data performs surprisingly well on a test set derived from $19 \times 19$ professional game data. Possible directions for further improvements are briefly discussed.

## 1 Introduction

Go is an ancient board game–over 3,000 years old [6, 5]–that poses unique opportunities and challenges for artificial intelligence and machine learning. The rules of Go are deceptively simple: two opponents alternatively place black and white stones on the empty intersections of an odd-sized square board, traditionally of size $19 \times 19$. The goal of the game, in simple terms, is for each player to capture as much territory as possible across the board by encircling the opponent's stones. This disarming simplicity, however, conceals a formidable combinatorial complexity [2]. On a $19 \times 19$ board, there are approximately $3^{19 \times 19} = 10^{172.24}$ possible board configurations and, on average, on the order of 200-300 possible moves at each step of the game, preventing any form of semi-exhaustive search. For comparison purposes, the game of chess has a much smaller branching factor, on the order of 35-40 [10, 7]. Today, computer chess programs, built essentially on search techniques and running on a simple PC, can rival or even surpass the best human players. In contrast, and in spite of several decades of significant research efforts and of progress in hardware speed, the best Go programs of today are easily defeated by an average human amateur.

Besides the intrinsic challenge of the game, and the non-trivial market created by over 100 million players worldwide, Go raises other important questions for our understanding of natural or artificial intelligence in the distilled setting created by the simple rules of a game, uncluttered by the endless complexities of the "real world". For example, to many observers, current computer solutions to chess appear "brute force", hence "unintelligent". But is this perception correct, or an illusion–is there something like true intelligence beyond "brute force" and computational power? Where is Go situated in the apparent tug-of-war between intelligence and sheer computational power?

Another fundamental question that is particularly salient in the Go setting is the question of knowledge transfer. Humans learn to play Go on boards of smaller sized–typically $9 \times 9$–and then "transfer" their knowledge to the larger $19 \times 19$ standard size. How can we develop algorithms that are capable of knowledge transfer?

Here we take modest steps towards addressing these challenges by developing a scalable machine learning approach to Go. Clearly good evaluation functions and search algorithms are essential ingredients of computer board-game systems. Here we focus primarily on the problem of learning a good evaluation function for Go in a scalable way. We do include simple search algorithms in our system, as many other programs do, but this is not the primary focus. By scalability we imply that a main goal is to develop a system more or less automatically, using machine learning approaches, with minimal human intervention and handcrafting. The system ought to be able to transfer information from one board size (e.g. $9 \times 9$), to another size (e.g. $19 \times 19$).

We take inspiration in three ingredients that seem to be essential to the Go human evaluation process: the understanding of local patterns, the ability to combine patterns, and the ability to relate tactical and strategic goals. Our system is built to learn these three capabilities automatically and attempts to combine the strengths of existing systems while avoiding some of their weaknesses. The system is capable of automatically learning the propensity of local patterns from a library of games. Propensity and other local tactical information are fed into a recursive neural network, derived from a Bayesian network architecture. The network integrates local information across the board and produces local outputs that represent local territory ownership probabilities. The aggregation of these probabilities provides an effective strategic evaluation function that is an estimate of the expected area at the end (or at other stages) of the game. Local area targets for training can be derived from datasets of human games. The main results we present here are derived on a $19 \times 19$ board using a player trained using only $9 \times 9$ game data.

## 2 Data

Because the approach to be described emphasizes scalability and learning, we are able to train our systems at a given board size and use it to play at different sizes, both larger and smaller. Pure bootstrap approaches to Go where computer players are initialized randomly and play large numbers of games, such as evolutionary approaches or reinforcement learning, have been tried [11]. We have implemented these approaches and used them for small board sizes $5 \times 5$ and $7 \times 7$. However, in our experience, these approaches do not scale up well to larger board sizes. For larger board sizes, better results are obtained using training data derived from records of games played by humans. We used available data at board sizes $9 \times 9$, $13 \times 13$, and $19 \times 19$.

**Data for** $9 \times 9$ **Boards:** This data consists of 3,495 games. We randomly selected 3,166 games (90.6%) for training, and the remaining 328 games (9.4%) for validation. Most of the games in this data set are played by amateurs. A subset of 424 games (12.13%) have at least one player with an olf ranking of 29, corresponding to a very good amateur player.

**Data for** $13 \times 13$ **Boards:** This data consists of 4175 games. Most of the games, however, are played by rather weak players and therefore cannot be used for training. For validation purposes, however, we retained a subset of 91 games where both players have an olf ranking greater or equal to 25–the equivalent of a good amateur player.

**Data for** $19 \times 19$ **Boards:** This high-quality data set consists of 1835 games played by professional players (at least 1 dan). A subset of 1131 games (61.6%) are played by 9 dan players (the highest possible ranking). This is the dataset used in [12].

## 3 System Architecture

### 3.1 Evaluation Function, Outputs, and Targets

Because Go is a game about territory, it is sensible to have "expected territory" be the evaluation function, and to decompose this expectation as a sum of local probabilities. More specifically, let $A_{ij}(t)$ denote the ownership of intersection $ij$ on the board at time $t$ during the game. At the end of a

game, each intersection can be black, white, or both [1]. Black is represented as 1, white as 0, and both as 0.5. The same scheme with 0.5 for empty intersections, or more complicated schemes, can be used to represent ownership at various intermediate stages of the game. Let $O_{ij}(t)$ be the output of the learning system at intersection $ij$ at time $t$ in the game. Likewise, let $T_{ij}(t)$ be the corresponding training target. In the most simple case, we can use $T_{ij}(t) = A_{ij}(T)$, where $T$ denotes the end of the game. In this case, the output $O_{ij}(t)$ can be interpreted as the probability $P_{ij}(t)$, estimated at time $t$, of owning the $ij$ intersection at the end of the game. Likewise, $\sum_{ij} O_{ij}(t)$ is the estimate, computed at time $t$, of the total expected area at the end of the game.

Propagation of information provided by targets/rewards computed at the end of the game only, however, can be problematic. With a dataset of training examples, this problem can be addressed because intermediary area values $A_{ij}(t)$ are available for training for any $t$. In the simulations presented here, we use a simple scheme

$$T_{ij}(t) = (1 - w)A_{ij}(T) + wA_{ij}(t + k) \tag{1}$$

$w \geq 0$ is a parameter that controls the convex combination between the area at the end of the game and the area at some step $t + k$ in the more near future. $w = 0$ corresponds to the simple case described above where only the area at the end of the game is used in the target function. Other ways of incorporating target information from intermediary game positions are discussed briefly at the end.

To learn the evaluation function and the targets, we propose to use a graphical model (Bayesian network) which in turn leads to a directed acyclic graph recursive neural network (DAG-RNN) architecture.

## 3.2 DAG-RNN Architectures

The architecture is closely related to an architecture originally proposed for a problem in a completely different area – the prediction of protein contact maps [8, 1]. As a Bayesian network, the architecture can be described in terms of the DAG in Figure 1 where the nodes are arranged in 6 lattice planes reflecting the Go board spatial organization. Each plane contains $N \times N$ nodes arranged on the vertices of a square lattice. In addition to the input and output planes, there are four hidden planes for the lateral propagation and integration of information across the Go board. Within each hidden plane, the edges of the quadratic lattice are oriented towards one of the four cardinal directions (NE, NW, SE, and SW). Directed edges within a column of this architecture are given in Figure 1b. Thus each intersection $ij$ in a $N \times N$ board is associated with six units. These units consist of an input unit $I_{ij}$, four hidden units $H_{ij}^{NE}, H_{ij}^{NW}, H_{ij}^{SW}, H_{ij}^{SE}$, and an output unit $O_{ij}$.

In a DAG-RNN the relationships between the variables are deterministic, rather than probabilistic, and implemented in terms of neural networks with weight sharing. Thus the previous architecture, leads to a DAG-RNN architecture consisting of 5 neural networks in the form

$$\begin{cases} O_{i,j} = \mathcal{N}_O(I_{i,j}, H_{i,j}^{NW}, H_{i,j}^{NE}, H_{i,j}^{SW}, H_{i,j}^{SE}) \\ H_{i,j}^{NE} = \mathcal{N}_{NE}(I_{i,j}, H_{i-1,j}^{NE}, H_{i,j-1}^{NE}) \\ H_{i,j}^{NW} = \mathcal{N}_{NW}(I_{i,j}, H_{i+1,j}^{NW}, H_{i,j-1}^{NW}) \\ H_{i,j}^{SW} = \mathcal{N}_{SW}(I_{i,j}, H_{i+1,j}^{SW}, H_{i,j+1}^{SW}) \\ H_{i,j}^{SE} = \mathcal{N}_{SE}(I_{i,j}, H_{i-1,j}^{SE}, H_{i,j+1}^{SE}) \end{cases} \tag{2}$$

where, for instance, $\mathcal{N}_O$ is a single neural network that is shared across all spatial locations. In addition, since Go is "isotropic" we use a single network shared across the four hidden planes. Go however involves strong boundaries effects and therefore we add one neural network $\mathcal{N}_C$ for the corners, shared across all four corners, and one neural network $\mathcal{N}_S$ for each side position, shared across all four sides. In short, the entire Go DAG-RNN architecture is described by four feedforward NNs (corner, side, lateral, output) that are shared at all corresponding locations. For each one of these feedforward neural networks, we have experimented with several architectures, but we

typically use a single hidden layer. The DAG-RNN in the main simulation results uses 16 hidden nodes and 8 output nodes for the lateral propagation networks, and 16 hidden nodes and one output node for the output network. All transfer functions are logistic. The total number of free parameters is close to 6000.

Because the underlying graph is acyclic, these networks can be unfolded in space and training can proceed by simple gradient descent (back-propagation) taking into account relevant symmetries and weight sharing. Networks trained at one board size can be reused at any other board size, providing a simple mechanism for reusing and extending acquired knowledge. For a board of size $N \times N$, the training procedure scales like $O(WMN^4)$ where $W$ is the number of adjustable weights, and $M$ is the number of training games. There are roughly $N^2$ board positions in a game and, for each position, $N^2$ outputs $O_{ij}$ to be trained, hence the $O(N^4)$ scaling. Both game records and the positions within each selected game record are randomly selected during training. Weights are updated essentially on line, once every 10 game positions. Training a single player on our $9 \times 9$ data takes on the order of a week on a current desktop computer, corresponding roughly to 50 training epochs at 3 hours per epoch.

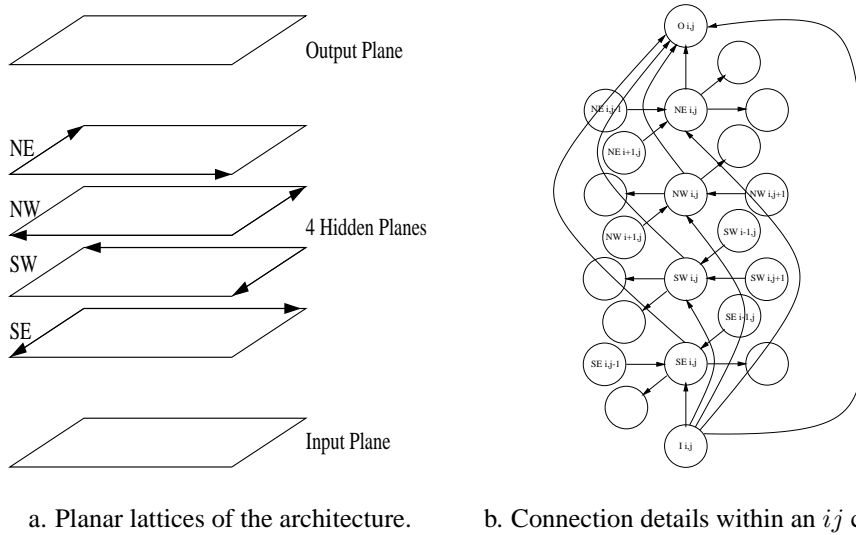

a. Planar lattices of the architecture.    b. Connection details within an $ij$ column.

Figure 1: (a) The nodes of a DAG-RNN are regularly arranged in one input plane, one output plane, and four hidden planes. In each plane, nodes are arranged on a square lattice. The hidden planes contain directed edges associated with the square lattices. All the edges of the square lattice in each hidden plane are oriented in the direction of one of the four possible cardinal corners: NE, NW, SW, and SE. Additional directed edges run vertically in column from the input plane to each hidden plane and from each hidden plane to the output plane. (b) Connection details within one column of Figure 1a. The input node is connected to four corresponding hidden nodes, one for each hidden plane. The input node and the hidden nodes are connected to the output node. $I_{ij}$ is the vector of inputs at intersection $ij$. $O_{ij}$ is the corresponding output. Connections of each hidden node to its lattice neighbors within the same plane are also shown.

### 3.3   Inputs

At a given board intersection, the input vector $I_{ij}$ has multiple components–listed in Table 1. The first three components–stone type, influence, and propensity–are associated with the corresponding intersection and a *fixed* number of surrounding locations. Influence and propensity are described below in more detail. The remaining features correspond to *group* properties involving variable numbers of neighboring stones and are self explanatory for those who are familiar with Go. The group $G_{ij}$ associated with a given intersection is the maximal set of stones of the same color that are connected to it. Neighboring (or connected) opponent groups of $G_{ij}$ are groups of the opposite color that are directly connected (adjacent) to $G_{ij}$. The idea of using higher order liberties is from Werf [13]. $O_{1st}$ and $O_{2nd}$ provide the number of true eyes and the number of liberties of the weakest and

the second weakest neighboring opponent groups. Weakness here is defined in alphabetical order with respect to the number of eyes first, followed by the number of liberties.

Table 1: Typical input features. The first three features–stone type, influence, and propensity–are properties associated with the corresponding intersection and a fixed number of surrounding locations. The other properties are group properties involving variable numbers of neighboring stones.

| Feature | Description |
| --- | --- |
| b,w,e | the stone type: black, white or empty |
| influence | the influence from the stones of the same color and the opposing color |
| propensity | a local statistics computed from $3 \times 3$ patterns in the training data (section 3.3) |
| $N_{eye}$ | the number of true eyes |
| $N_{1st}$ | the number of liberties, which is the number of empty intersections connected to a group of stones. We also call it the 1st-order liberties |
| $N_{2nd}$ | the number of 2nd-order liberties, which is defined as the liberties of the 1st-order liberties |
| $N_{3rd}$ | the number of 3rd-order liberties, which is defined as the liberties of the 2nd-order liberties |
| $N_{4th}$ | the number of 4th-order liberties, which is defined as the liberties of the 3rd-order liberties |
| $O_{1st}$ | features of the weakest connected opponent group (stone type, number of liberties, number of eyes) |
| $O_{2nd}$ | features of the second weakest connected opponent group (stone type, number of liberties, number of eyes) |

**Influence:** We use two types of influence calculation. Both algorithms are based on Chen's method [4]. One is an exact implementation of Chen's method. The other uses a stringent influence propagation rule. In Chen's exact method, any opponent stone can block the propagation of influence. With a stringent influence propagation rule, an opponent stone can block the propagation of influence if and only if it is stronger than the stone emitting the influence. Strength is again defined in alphabetical order with respect to the number of eyes first, followed by the number of liberties.

**Propensity–Automated Learning and Scoring of a Pattern Library:** We develop a method to learn local patterns and their value automatically from a database of games. The basic method is illustrated in the case of $3 \times 3$ patterns, which are used in the simulations. Considering rotation and mirror symmetries, there are 10 unique locations for a $3 \times 3$ window on a $9 \times 9$ board (see also [9]). Given any $3 \times 3$ pattern of stones on the board and a set of games, we then compute nine numbers, one for each intersection. These numbers are local indicators of strength or propensity. The propensity $S_{ij}(p)$ of each intersection $ij$ associated with stone pattern $p$ and a $3 \times 3$ window $w$ is defined as:

$$S_{ij}^{w}(p) = \frac{NB_{ij}(p) - NW_{ij}(p)}{NB_{ij}(p) + NW_{ij}(p) + C} \qquad (3)$$

where $NB_{ij}(p)$ is the number of times that pattern $p$ ends with a black stone at intersection $ij$ at the end of the games in the data, and $NW_{ij}(p)$ is the same for a white stone. Both $NB_{ij}(p)$ and $NW_{ij}(p)$ are computed taking into account the location and the symmetries of the corresponding window $w$. $C$ plays a regularizing role in the case of rare patterns and is set to 1 in the simulations. Thus $S_{ij}^{w}(p)$ is an empirical normalized estimate of the local differential propensity towards conquering the corresponding intersection in the local context provided by the corresponding pattern and window.

In general, a given intersection $ij$ on the board is covered by several $3 \times 3$ windows. Thus, for a given intersection $ij$ on a given board, we can compute a value $S_{ij}^{w}(p)$ for each different window that contains the intersection. In the following simulations, a single final value $S_{ij}(p)$ is computed by averaging over the different $w$'s. However, more complex schemes that retain more information can easily be envisioned by, for instance: (1) computing also the standard deviation of the $S_{ij}^{w}(p)$ as a function of $w$; (2) using a weighted average, weighted by the importance of the window $w$; and (3) using the entire set of $S_{ij}^{w}(p)$ values, as $w$ varies around $ij$, to augment the input vector.

## 3.4 Move Selection and Search

For a given position, the next move can be selected using one-level search by considering all possible legal moves and computing the estimate at time $t$ of the total expected area $E = \sum_{ij} O_{ij}(t)$ at the end of the game, or some intermediate position, or a combination of both, where $O_{ij}(t)$ are the outputs (predicted probabilities) of the DAG-RNNs. The next move can be chosen by maximizing this evaluation function (1-ply search). Alternatively, Gibbs sampling can be used to choose the next move among all the legal moves with a probability proportional to $e^{E/Temp}$, where $Temp$ is a temperature parameter [3, 11, 12]. We have also experimented with a few other simple search schemes, such as 2-ply search (MinMax).

## 4 Results

We trained a large number of players using the methods described above. In the absence of training data, we used pure bootstrap approaches (e.g. reinforcement learning) at sizes $5 \times 5$ and $7 \times 7$ with results that were encouraging but clearly insufficient. Not surprisingly, when used to play at larger board sizes, the RNNs trained at these small board sizes yield rather weak players. The quality of most $13 \times 13$ games available to us is too poor for proper training, although a small subset can be used for validation purposes. We do not have any data for sizes $N = 11, 15$, and $17$. And because of the $O(N^4)$ scaling, training systems directly at $19 \times 19$ takes many months and is currently in progress. Thus the most interesting results we report are derived by training the RNNs using the $9 \times 9$ game data, and using them to play at $9 \times 9$ and, more importantly, at larger board sizes. Several $9 \times 9$ players achieve top comparable performance. For conciseness, here we report the results obtained with one of them, trained with target parameters $w = 0.25$ and $k = 2$ in Equation 1,

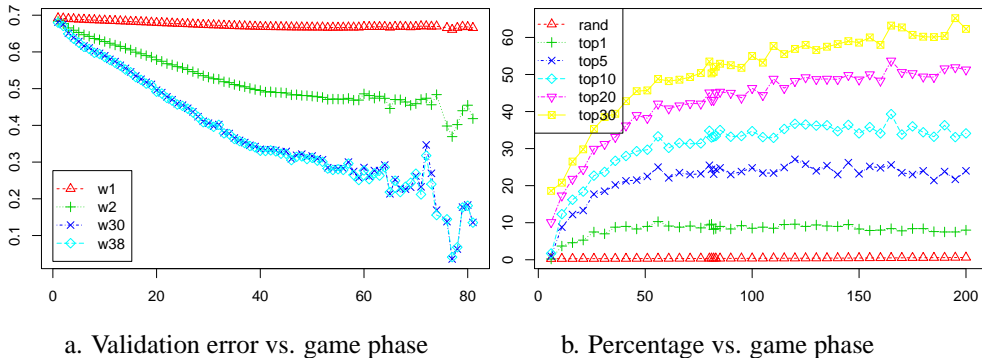

a. Validation error vs. game phase      b. Percentage vs. game phase

Figure 2: (a) Validation error vs. game phase. Phase is defined by the total number of stones on the board. The four curves respectively represent the validation errors of the neural network after 1, 2, 33, and 38 epochs of training. (b) Percentage of moves made by professional human players on boards of size $19 \times 19$ that are contained in the $m$ top-ranked moves according to the DAG-RNN trained on $9 \times 9$ amateur data, for various values of $m$. The baseline associated with the red curve corresponds to a random uniform player.

Figure 2a shows how the validation error changes as training progresses. Validation error here is defined as the relative entropy between the output probabilities produced by the RNN and the target probabilities, computed on the validation data. The validation error decreases quickly during the first epochs. In this case, no substantial decrease in validation error is observed after epoch 30. Note also how the error is smaller towards the end of the game due both to the reduction in the number of possible moves and the strong end-of-game training signal.

An area and hence a probability can be assigned by the DAG-RNN to each move, and used to rank them, as described in section 3.4. Thus we can compute the average probability of moves played by good human players according to the DAG-RNN or other probabilistic systems such as [12]. In Table 2, we report such probabilities for several systems and at different board sizes. For size $19 \times 19$, we use the same test set used in [12]. Boltzmann5 and BoltzmannLiberties are their results reported in the pre-published version of their NIPS paper. At this size, the probabilities in

Table 2: Probabilities assigned by different systems to moves played by human players in test data.

| Board Size | System | Log Probability | Probability |
|---|---|---|---|
| $9 \times 9$ | Random player | -4.13 | 1/62 |
| $9 \times 9$ | RNN(1-ply search) | -1.86 | 1/7 |
| $13 \times 13$ | Random player | -4.88 | 1/132 |
| $13 \times 13$ | RNN(1-ply search) | -2.27 | 1/10 |
| $19 \times 19$ | Random player | -5.64 | 1/281 |
| $19 \times 19$ | Boltzmann5 | -5.55 | 1/254 |
| $19 \times 19$ | BoltzmannLiberties | -5.27 | 1/194 |
| $19 \times 19$ | RNN(1-ply search) | -2.70 | **1/15** |

the table are computed using the 80-83rd moves of each game. For boards of size $19 \times 19$, a random player that selects moves uniformly at random among legal moves assigns a probability of 1/281 to the moves played by professional players in the data set. BoltzmannLiberties was able to improve this probability to 1/194. Our best DAG-RNNs trained using amateur data at $9 \times 9$ are capable of bringing this probability further down to 1/15 (also a considerable improvement over our previous 1/42 performance presented in April 2006 at the Snowbird Learning Conference). A remarkable example where the top ranked move according to the DAG-RNN coincides with the move actually played in a game between two very highly-ranked players is given in Figure 3, illustrating also the underlying probabilistic territory calculations.

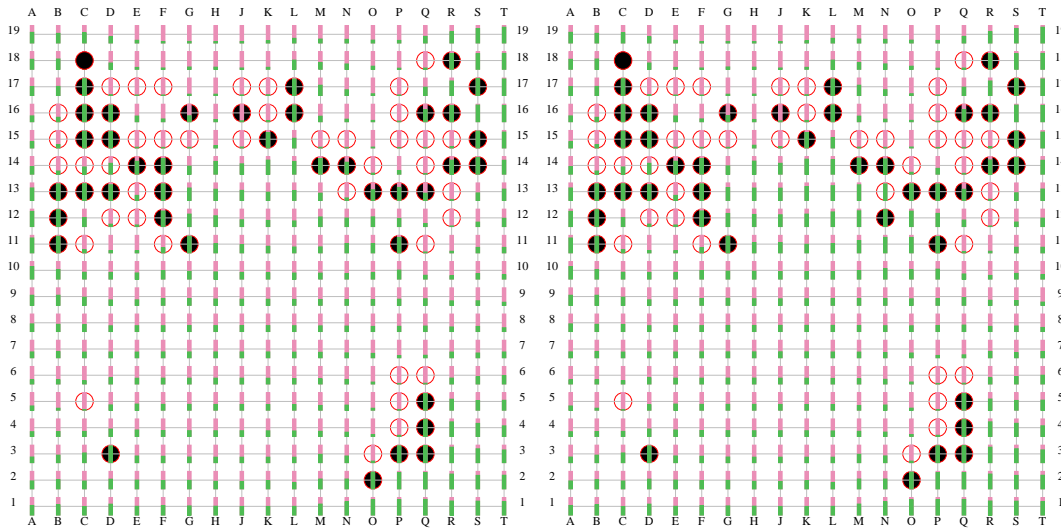

Figure 3: Example of an outstanding move based on territory predictions made by the DAG-RNN. For each intersection, the height of the green bar represents the estimated probability that the intersection will be owned by black at the end of the game. The figure on the left shows the predicted probabilities if black passes. The figure on the right shows the predicted probabilities if black makes the move at N12. N12 causes the greatest increase in green area and is top-ranked move for the DAG-RNN. Indeed this is the move selected in the game played by Zhou, Heyang (black, 8 dan) and Chang, Hao (white, 9 dan) on 10/22/2000.

Figure 2b, provides a kind of ROC curve by displaying the percentage of moves made by professional human player on boards of size $19 \times 19$ that are contained in the $m$ top-ranked moves according to the DAG-RNN trained on $9 \times 9$ amateur data, for various values of $m$ across all phases of the game. For instance, when there are 80 stones on the board, and hence on the order of 300 legal moves available, there is a 50% chance that a move selected by a very highly ranked human player (dan 9) is found among the top 30 choices produced by the DAG-RNN.

# 5 Conclusion

We have designed a DAG-RNN for the game of Go and demonstrated that it can learn territory predictions fairly well. Systems trained using only a set of $9 \times 9$ amateur games achieve surprisingly good performance on a $19 \times 19$ test set that contains 1835 professional played games. The methods and results presented clearly point also to several possible direction of improvement that are currently under active investigation. These include: (1) obtaining larger data sets and training systems of size greater than $9 \times 9$; (2) exploiting patterns that are larger than $3 \times 3$, especially at the beginning of the game when the board is sparsely occupied and matching of large patterns is possible using, for instance, Zobrist hashing techniques [14]; (3) combining different players, such as players trained at different board sizes, or players trained on different phases of the game; and (4) developing better, non-exhaustive but deeper, search methods.

**Acknowledgments**

The work of PB and LW has been supported by a Laurel Wilkening Faculty Innovation award and awards from NSF, BREP, and Sun Microsystems to PB. We would like to thank Jianlin Chen for developing a web-based Go graphical user interface, Nicol Schraudolph for providing the $9 \times 9$ and $13 \times 13$ data, and David Stern for providing the $19 \times 19$ data.

## Footnotes

[1]This is called "seki". Seki is a situation where two live groups share liberties and where neither of them can fill them without dying.

## References

[1] P. Baldi and G. Pollastri. The principled design of large-scale recursive neural network architectures–DAG-RNNs and the protein structure prediction problem. *Journal of Machine Learning Research*, 4:575–602, 2003.

[2] E. Berlekamp and D. Wolfe. *Mathematical Go–Chilling gets the last point*. A K Peters, Wellesley, MA, 1994.

[3] B. Brugmann. Monte Carlo Go. 1993. URL: `ftp://www.joy.ne.jp/welcome/igs/Go/computer/mcgo.tex.Z`.

[4] Zhixing Chen. Semi-empirical quantitative theory of Go part 1: Estimation of the influence of a wall. *ICGA Journal*, 25(4):211–218, 2002.

[5] W. S. Cobb. *The Book of GO*. Sterling Publishing Co., New York, NY, 2002.

[6] K. Iwamoto. *GO for Beginners*. Pantheon Books, New York, NY, 1972.

[7] Aske Plaat, Jonathan Schaeffer, Wim Pijls, and Arie de Bruin. Exploiting graph properties of game trees. In *13th National Conference on Artificial Intelligence (AAAI'96)*, pages 234–239. 1996.

[8] G. Pollastri and P. Baldi. Prediction of contact maps by GIOHMMs and recurrent neural networks using lateral propagation from all four cardinal corners. *Bioinformatics*, 18:S62–S70, 2002.

[9] Liva Ralaivola, Lin Wu, and Pierre Balid. SVM and Pattern-Enriched Common Fate Graphs for the game of Go. *ESANN 2005*, 27-29:485–490, 2005.

[10] Stuart J. Russell and Peter Norvig. *Artificial Intelligence: A Modern Approach*. Prentice Hall, 2nd edition, 2002.

[11] N. N. Schrauldolph, P. Dayan, and T. J. Sejnowski. Temporal difference learning of position evaluation in the game of Go. In *Advances in Neural Information Processing Systems 6*, pages 817–824. 1994.

[12] David H. Stern, Thore Graepel, and David J. C. MacKay. Modelling uncertainty in the game of Go. In *Advances in Neural Information Processing Systems 17*, pages 1353–1360. 2005.

[13] E. Werf, H. Herik, and J. Uiterwijk. Learning to score final positions in the game of Go. In *Advances in Computer Games: Many Games, Many Challenges*, pages 143–158. 2003.

[14] Albert L. Zobrist. A new hashing method with application for game playing. 1970. Technical report 88, University of Wisconsin, April 1970. Reprinted in ICCA Journal, 13(2), (1990), pp. 69-73.
